# Context-Sensitive Decision Forests for Object Detection

**Peter Kontschieder**[1]  **Samuel Rota Bulò**[2]  **Antonio Criminisi**[3]
**Pushmeet Kohli**[3]  **Marcello Pelillo**[2]  **Horst Bischof**[1]
[1]ICG, Graz University of Technology, Austria
[2]DAIS, Università Ca' Foscari Venezia, Italy
[3]Microsoft Research Cambridge, UK

## Abstract

In this paper we introduce Context-Sensitive Decision Forests - A new perspective to exploit contextual information in the popular decision forest framework for the object detection problem. They are tree-structured classifiers with the ability to access intermediate prediction (here: classification and regression) information during training and inference time. This intermediate prediction is available for each sample and allows us to develop context-based decision criteria, used for refining the prediction process. In addition, we introduce a novel split criterion which in combination with a priority based way of constructing the trees, allows more accurate regression mode selection and hence improves the current context information. In our experiments, we demonstrate improved results for the task of pedestrian detection on the challenging TUD data set when compared to state-of-the-art methods.

## 1 Introduction and Related Work

In the last years, the random forest framework [1, 6] has become a very popular and powerful tool for classification and regression problems by exhibiting many appealing properties like inherent multi-class capability, robustness to label noise and reduced tendencies to overfitting [7]. They are considered to be close to an ideal learner [13], making them attractive in many areas of computer vision like image classification [5, 17], clustering [19], regression [8] or semantic segmentation [24, 15, 18]. In this work we show how the decision forest algorithm can be extended to include contextual information during learning and inference for classification and regression problems.

We focus on applying random forests to object detection, *i.e.* the problem of localizing multiple instances of a given object class in a test image. This task has been previously addressed in random forests [9], where the trees were modified to learn a mapping between the appearance of an image patch and its relative position to the object category centroid (*i.e.* center voting information). During inference, the resulting *Hough Forest* not only performs classification on test samples but also casts probabilistic votes in a generalized Hough-voting space [3] that is subsequently used to obtain object center hypotheses. Ever since, a series of applications such as tracking and action recognition [10], body-joint position estimation [12] and multi-class object detection [22] have been presented. However, Hough Forests typically produce non-distinctive object hypotheses in the Hough space and hence there is the need to perform non-maximum suppression (NMS) for obtaining the final results. While this has been addressed in [4, 26], another shortcoming is that standard (Hough) forests treat samples in a completely independent way, *i.e.* there is no mechanism that encourages the classifier to perform consistent predictions.

Within this work we are proposing that context information can be used to overcome the aforementioned problems. For example, training data for visual learning is often represented by images in form of a (regular) pixel grid topology, *i.e.* objects appearing in natural images can often be found in a specific context. The importance of contextual information was already highlighted in the 80's with

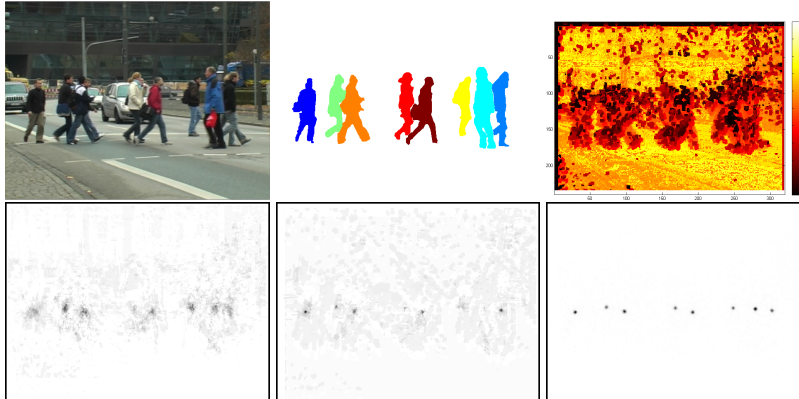

Figure 1: Top row: Training image, label image, visualization of priority-based growing of tree (the lower, the earlier the consideration during training.). Bottom row: Inverted Hough image using [9] and breadth-first training after 6 levels ($2^6 = 64$ nodes), Inverted Hough image after growing 64 nodes using our priority queue, Inverted Hough image using priority queue shows distinctive peaks at the end of training.

a pioneering work on relaxation labelling [14] and a later work with focus on inference tasks [20] that addressed the issue of learning within the same framework. More recently, contextual information has been used in the field of object class segmentation [21], however, mostly for high-level reasoning in random field models or to resolve contradicting segmentation results. The introduction of contextual information as additional features in low-level classifiers was initially proposed in the Auto-context [25] and Semantic Texton Forest [24] models. Auto-context shows a general approach for classifier boosting by iteratively learning from appearance and context information. In this line of research [18] augmented the feature space for an *Entanglement Random Forest* with a classification feature, that is consequently refined by the class posterior distributions according to the progress of the trained subtree. The training procedure is allowed to perform tests for specific, contextual label configurations which was demonstrated to significantly improve the segmentation results. However, the

In this paper we are presenting *Context-Sensitve Decision Forests* - A novel and unified interpretation of Hough Forests in light of contextual sensitivity. Our work is inspired by Auto-Context and Entanglement Forests, but instead of providing only posterior *classification* results from an earlier level of the classifier construction during learning and testing, we additionally provide *regression (voting)* information as it is used in Hough Forests. The second core contribution of our work is related to how we grow the trees: Instead of training them in a depth- or breadth-first way, we propose a priority-based construction (which could actually consider depth- or breadth-first as particular cases). The priority is determined by the current training error, *i.e.* we first grow the parts of the tree where we experience higher error. To this end, we introduce a unified splitting criterion that estimates the joint error of classification and regression. The consequence of using our priority-based training are illustrated in Figure 1: Given the training image with corresponding label image (top row, images 1 and 2), the tree first tries to learn the foreground samples as shown in the color-coded plot (top row, image 3, colors correspond to index number of nodes in the tree). The effects on the intermediate prediction quality are shown in the bottom row for the regression case: The first image shows the regression quality after training a tree with 6 levels ($2^6 = 64$ nodes) in a breadth-first way while the second image shows the progress after growing 64 nodes according to the priority based training. Clearly, the modes for the center hypotheses are more distinctive which in turn yields to more accurate intermediate regression information that can be used for further tree construction. Our third contribution is a new family of split functions that allows to learn from training images containing multiple training instances as shown for the pedestrians in the example. We introduce a test that checks the centroid compatibility for pairs of training samples taken from the context, based on the intermediate classification and regression derived as described before. To assess our contributions, we performed several experiments on the challenging TUD pedestrian data set [2], yielding a significant improvement of $9\%$ in the recall at $90\%$ precision rate in comparison to standard Hough Forests, when learning from crowded pedestrian images.

## 2 Context-Sensitive Decision Trees

This section introduces the general idea behind the context-sensitive decision forest without references to specific applications. Only in Section 3 we show a particular application to the problem of object detection. After showing some basic notational conventions that are used in the paper, we provide a section that revisits the random forest framework for classification and regression tasks from a joint perspective, *i.e.* a theory allowing to consider *e.g.* [1, 11] and [9] in a unified way. Starting from this general view we finally introduce the context-sensitive forests in 2.2.

**Notations.** In the paper we denote *vectors* using boldface lowercase (*e.g.* $\mathbf{d}$, $\mathbf{u}$, $\mathbf{v}$) and *sets* by using uppercase calligraphic (*e.g.* $\mathcal{X}$, $\mathcal{Y}$) symbols. The sets of real, natural and integer numbers are denoted with $\mathbb{R}$, $\mathbb{N}$ and $\mathbb{Z}$ as usually. We denote by $2^{\mathcal{X}}$ the power set of $\mathcal{X}$ and by $\mathbb{1}\left[P\right]$ the indicator function returning 1 or 0 according to whether the proposition $P$ is true or false. Moreover, with $\mathbb{P}(\mathcal{Y})$ we denote the set of probability distributions having $\mathcal{Y}$ as sample space and we implicitly assume that some $\sigma$-algebra is defined on $\mathcal{Y}$. We denote by $\delta(x)$ the Dirac delta function. Finally, $\mathbb{E}_{x\sim Q}\left[f(x)\right]$ denotes the expectation of $f(x)$ with respect to $x$ sampled according to distribution $Q$.

### 2.1 Random Decision Forests for joint classification and regression

A (binary) *decision tree* is a tree-structured predictor[1] where, starting from the root, a sample is routed until it reaches a leaf where the prediction takes place. At each internal node of the tree the decision is taken whether the sample should be forwarded to the left or right child, according to a binary-valued function. In formal terms, let $\mathcal{X}$ denote the input space, let $\mathcal{Y}$ denote the output space and let $\mathcal{T}^{dt}$ be the set of decision trees. In its simplest form a decision tree consists of a single node (a *leaf*) and is parametrized by a probability distribution $Q \in \mathbb{P}(\mathcal{Y})$ which represents the posterior probability of elements in $\mathcal{Y}$ given any data sample reaching the leaf. We denote this (admittedly rudimentary) tree as $\text{LF}\left(Q\right) \in \mathcal{T}^{td}$. Otherwise, a decision tree consists of a node with a left and a right sub-tree. This node is parametrized by a *split function* $\phi : \mathcal{X} \to \{0, 1\}$, which determines whether to route a data sample $x \in \mathcal{X}$ reaching it to the left decision sub-tree $t_l \in \mathcal{T}^{dt}$ (if $\phi(x) = 0$) or to the right one $t_r \in \mathcal{T}^{dt}$ (if $\phi(x) = 1$). We denote such a tree as $\text{ND}\left(\phi, t_l, t_r\right) \in \mathcal{T}^{td}$. Finally, a *decision forest* is an ensemble $\mathcal{F} \subseteq \mathcal{T}^{td}$ of decision trees which makes a prediction about a data sample by averaging over the single predictions gathered from all trees.

**Inference.** Given a decision tree $t \in \mathcal{T}^{dt}$, the associated posterior probability of each element in $\mathcal{Y}$ given a sample $x \in \mathcal{X}$ is determined by finding the probability distribution $Q$ parametrizing the leaf that is reached by $x$ when routed along the tree. This is compactly presented with the following definition of $P(y|x,t)$, which is inductive in the structure of $t$:

$$P(y \,|\, x, t) = \begin{cases} Q(y) & \text{if } t = \text{LF}\left(Q\right) \\ P(y \,|\, x, t_l) & \text{if } t = \text{ND}\left(\phi, t_l, t_r\right) \text{ and } \phi(x) = 0 \\ P(y \,|\, x, t_r) & \text{if } t = \text{ND}\left(\phi, t_l, t_r\right) \text{ and } \phi(x) = 1 \,. \end{cases} \tag{1}$$

Finally, the combination of the posterior probabilities derived from the trees in a forest $\mathcal{F} \subseteq \mathcal{T}^{dt}$ can be done by an averaging operation [6], yielding a single posterior probability for the whole forest:

$$P(y|x, \mathcal{F}) = \frac{1}{|\mathcal{F}|} \sum_{t \in \mathcal{F}} P(y|x, t) \,. \tag{2}$$

**Randomized training.**

A random forest is created by training a set of random decision trees independently on random subsets of the training data $\mathcal{D} \subseteq \mathcal{X} \times \mathcal{Y}$. The training procedure for a single decision tree heuristically optimizes a set of parameters like the tree structure, the split functions at the internal nodes and the density estimates at the leaves in order to reduce the prediction error on the training data. In order to prevent overfitting problems, the search space of possible split functions is limited to a random set and a minimum number of training samples is required to grow a leaf node. During the training procedure, each new node is fed with a set of training samples $\mathcal{Z} \subseteq \mathcal{D}$. If some stopping condition holds, depending on $\mathcal{Z}$, the node becomes a leaf and a density on $\mathcal{Y}$ is estimated based on $\mathcal{Z}$. Otherwise, an internal node is grown and a split function is selected from a pool of random ones in a way to minimize some sort of training error on $\mathcal{Z}$. The selected split function induces a partition

of $\mathcal{Z}$ into two sets, which are in turn becoming the left and right childs of the current node where the training procedure is continued, respectively.

We will now write this training procedure in more formal terms. To this end we introduce a function $\pi(\mathcal{Z}) \in \mathbb{P}(\mathcal{Y})$ providing a density on $\mathcal{Y}$ estimated from the training data $\mathcal{Z} \subseteq \mathcal{D}$ and a *loss function* $L(\mathcal{Z} \,|\, Q) \in \mathbb{R}$ penalizing wrong predictions on the training samples in $\mathcal{Z}$, when predictions are given according to a distribution $Q \in \mathbb{P}(\mathcal{Y})$. The loss function $L$ can be further decomposed in terms of a loss function $\ell(\cdot|Q) \,:\, \mathcal{Y} \to \mathbb{R}$ acting on each sample of the training set:

$$L(\mathcal{Z} \,|\, Q) = \sum_{(x,y) \in \mathcal{Z}} \ell(y \,|\, Q) \,. \tag{3}$$

Also, let $\Phi(\mathcal{Z})$ be a set of split functions randomly generated for a training set $\mathcal{Z}$ and given a split function $\phi \in \Phi(\mathcal{Z})$, we denote by $\mathcal{Z}_l^{\phi}$ and $\mathcal{Z}_r^{\phi}$ the sets identified by splitting $\mathcal{Z}$ according to $\phi$, *i.e.*

$$\mathcal{Z}_l^{\phi} = \{(x,y) \in \mathcal{Z} \,:\, \phi(x) = 0\} \qquad \text{and} \qquad \mathcal{Z}_r^{\phi} = \{(x,y) \in \mathcal{Z} \,:\, \phi(x) = 1\} \,.$$

We can now summarize the training procedure in terms of a recursive function $g \,:\, 2^{\mathcal{X} \times \mathcal{Y}} \to \mathcal{T}$, which generates a random decision tree from a training set given as argument:

$$g(\mathcal{Z}) = \begin{cases} \mathrm{LF}\left(\pi(\mathcal{Z})\right) & \text{if some stopping condition holds} \\ \mathrm{ND}\left(\phi, g(\mathcal{Z}_l^{\phi}), g(\mathcal{Z}_r^{\phi})\right) & \text{otherwise} \,. \end{cases} \tag{4}$$

Here, we determine the optimal split function $\phi$ in the pool $\Phi(\mathcal{Z})$ as the one minimizing the loss we incur as a result of the node split:

$$\phi \in \arg\min\left\{L(\mathcal{Z}_l^{\phi'}) + L(\mathcal{Z}_r^{\phi'}) \,:\, \phi' \in \Phi(\mathcal{Z})\right\} \tag{5}$$

where we compactly write $L(\mathcal{Z})$ for $L(\mathcal{Z}|\pi(\mathcal{Z}))$, *i.e.* the loss on $\mathcal{Z}$ obtained with predictions driven by $\pi(\mathcal{Z})$. A typical split function selection criterion commonly adopted for classification and regression is information gain. The equivalent counterpart in terms of loss can be obtained by using a log-loss, *i.e.* $\ell(y|Q) = -\log(Q(y))$. A further widely used criterion is based on Gini impurity, which can be expressed in this setting by using $\ell(y|Q) = 1 - Q(y)$.

Finally, the stopping condition that is used in (4) to determine whether to create a leaf or to continue branching the tree typically consists in checking $|\mathcal{Z}|$, *i.e.* the number of training samples at the node, or the loss $L(\mathcal{Z})$ are below some given thresholds, or if a maximum depth is reached.

## 2.2 Context-sensitive decision forests

A context-sensitive (CS) decision tree is a decision tree in which split functions are enriched with the ability of testing contextual information of a sample, before taking a decision about where to route it. We generate contextual information at each node of a decision tree by exploiting a truncated version of the same tree as a predictor. This idea is shared with [18], however, we introduce some novelties by tackling both, classification and regression problems in a joint manner and by leaving a wider flexibility in the tree truncation procedure. We denote the set of CS decision trees as $\mathcal{T}$. The main differences characterizing a CS decision tree $t \in \mathcal{T}$ compared with a standard decision tree are the following: a) *every* node (leaves and internal nodes) of $t$ has an associated probability distribution $Q \in \mathbb{P}(\mathcal{Y})$ representing the posterior probability of an element in $\mathcal{Y}$ given any data sample reaching it; b) internal nodes are indexed with distinct natural numbers $n \in \mathbb{N}$ in a way to preserve the property that children nodes have a larger index compared to their parent node; c) the split function at each internal node, denoted by $\varphi(\cdot|t') \,:\, \mathcal{X} \to \{0,1\}$, is bound to a CS decision tree $t' \in \mathcal{T}$, which is a truncated version of $t$ and can be used to compute intermediate, contextual information.

Similar to Section 2.1 we denote by $\mathrm{LF}\left(Q\right) \in \mathcal{T}$ the simplest CS decision tree consisting of a single leaf node parametrized by the distribution $Q$, while we denote by $\mathrm{ND}\left(n, Q, \varphi, t_l, t_r\right) \in \mathcal{T}$, the rest of the trees consisting of a node having a left and a right sub-tree, denoted by $t_l, t_r \in \mathcal{T}$ respectively, and being parametrized by the index $n$, a probability distribution $Q$ and the split function $\varphi$ as described above.

As shown in Figure 2, the truncation of a CS decision tree at each node is obtained by exploiting the indexing imposed on the internal nodes of the tree. Given a CS decision tree $t \in \mathcal{T}$ and $m \in \mathbb{N}$,

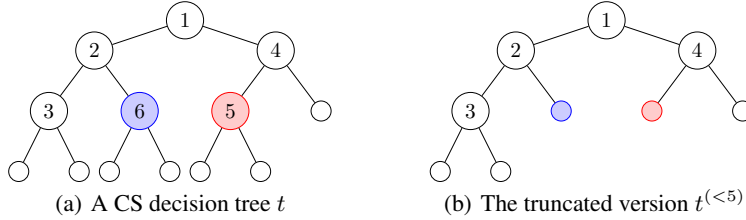

(a) A CS decision tree $t$         (b) The truncated version $t^{(<5)}$

Figure 2: On the left, we find a CS decision tree $t$, where only the internal nodes are indexed. On the right, we see the truncated version $t^{(<5)}$ of $t$, which is obtained by converting to leaves all nodes having index $\geq 5$ (we marked with colors the corresponding node transformations).

we denote by $t^{(<m)}$ a CS decision tree derived from $t$ in which only the internal nodes having index $< m$ are kept and the internal nodes with index $\geq m$ having a parent with index $< m$, or being the root node, are converted into leaves. Finally, all nodes left-over are pruned away.

**Inference.** The inference process, given a CS decision tree $t \in \mathcal{T}$, is equivalent to the one introduced for standard decision trees, with the only difference that a split function in a node indexed by $n$ can use the truncated version of the same decision tree $t^{(<n)}$ to additionally exploit contextual information while taking decisions about where to route samples. In the specific, the posterior probability of $y \in \mathcal{Y}$ given a sample $x \in \mathcal{X}$ is inductively defined as:

$$P(y \mid x, t) = \begin{cases} Q(y) & \text{if } t = \text{LF}(Q) \\ P(y \mid x, t_l) & \text{if } t = \text{ND}(n, \cdot, \varphi, t_l, t_r) \text{ and } \varphi(x \mid t^{(<n)}) = 0 \\ P(y \mid x, t_r) & \text{if } t = \text{ND}(n, \cdot, \varphi, t_l, t_r) \text{ and } \varphi(x \mid t^{(<n)}) = 1. \end{cases} \quad (6)$$

The same posterior probabilities with respect to a forest $\mathcal{F} \subseteq \mathcal{T}$ can be obtained as in (2).

**Prioritized node training.** The training process for CS decision forests consists in training an ensemble of CS decision trees independently on random subsets of the training set $\mathcal{D} \subseteq \mathcal{X} \times \mathcal{Y}$. Each CS decision tree is trained in an iterative way and, similar to the case of standard decision trees, a decision about whether to branch new nodes or produce a leaf is taken based on a subset of the training samples $\mathcal{Z} \subseteq \mathcal{D}$. However, in contrast to the standard setting, the learning process depends on the order in which nodes of the tree are grown because split functions depend on $t^{(<m)}$ which in turn is affected by the node ordering. In other words, we impose an explicit ordering on the recursive calls of function $g$ described in Equation (4). This ordering is determined by means of a priority queue, where the priority associated to each function call is determined according to a cost value. This cost can for instance be the depth at which a new node will be grown by the recursive call, in which case we enforce a breadth-first ordering, or the negative loss $-L(\mathcal{Z})$ defined as in (3), $\mathcal{Z}$ being the subset of the training data argument of the function call. This second option is particularly interesting because it forces the tree to split first the nodes where the training error measured in terms of the loss function is the highest. This indeed allows to reduce the uncertainty uniformly during the tree growth and in turn results in more reliable contextual information.

Whenever a new node is grown, it takes the time at which it was extracted from the priority queue as index. It is easy to see that the indexing deriving from this procedure never violates the property that children of a node have an index larger than the parent node. The split function selection is performed according to (5), the only difference being the type of split functions that are generated, which can exploit $t^{(<m)}$ to test contextual information.

## 3 Application to Object Detection

In this section we employ the CS decision trees for the problem of object detection, following a solution setting similar to [9]. Specifically, we adopt a patch-based abstraction of an image and the aim of the tree-based predictor is to jointly predict, for each patch, the foreground/background class it belongs to and a displacement vector pointing to the object's center. By collecting all the object position hypotheses from all foreground patches, we can setup a Hough space in which objects can be detected from the vote modes.

An image $I : \mathbb{Z}^2 \to \mathcal{F}$ is a function mapping pixels to elements of a feature space $\mathcal{F}$. The feature space here may include a variety of image cues, like color information, gradients, filter bank

responses, *etc.* . We denote by $I(\mathbf{u}) \in \mathcal{F}$ the feature vector associated to pixel $\mathbf{u}$ and by $\mathcal{I}$ the set of images, and by $I(\mathbf{u})_k$ the $k$th element of the feature vector associated to $\mathbf{u}$. The input space $\mathcal{X}$ for our learning problem is a set of patches, each represented as a pair $(\mathbf{u}, I) \in \mathbb{Z}^2 \times \mathcal{I}$, pixel $\mathbf{u}$ being the center of the patch in image $I$.

The output space $\mathcal{Y}$ is a set of pairs $(c, \mathbf{d})$, where $c \in \{0, 1\}$ is a binary class label indicating the presence of an object and $\mathbf{d} \in \mathbb{Z}^2$ is the displacement of the object's center. Hence, if a training sample $(\mathbf{u}, I) \in \mathcal{X}$ has $(c, \mathbf{d}) \in \mathcal{Y}$ as the ground-truth prediction then we have in image $I$ at location $\mathbf{u}$ either a background pixel ($c = 0$) or a foreground pixel ($c = 1$), *i.e.* belonging to an object and, if the second case holds, $\mathbf{u} + \mathbf{d}$ is the center of the object to which the pixel belongs. Note that $\mathcal{Y}$ encodes both the classification and regression part of the object detection task.

The loss function $\ell(c, \mathbf{d}|Q)$ that we employ for the computation of $L(\mathcal{Z}|Q)$ in (3) is given by

$$\ell(c, \mathbf{d} \,|\, Q) = \mathbb{E}_{(c', \mathbf{d}') \sim Q} \left[ \mathbb{1} \left[ c \neq c' \right] + \mathbb{1} \left[ (c, c') = (1, 1) \right] (1 - K_\sigma(\mathbf{d} - \mathbf{d}')) \right] \qquad (7)$$

where $K_\sigma(\mathbf{x}) = \exp(-\|\mathbf{x}\|^2/\sigma^2)$. This quantity measures the expected loss that we incur by predicting $(c', \mathbf{d}')$ in place of $(c, \mathbf{d})$, where $(c', \mathbf{d}')$ is sampled according to $Q$. The term under expectation behaves as a $0/1$ loss for all combinations of class labels, excepting the case $c = c' = 1$ where also the correct prediction of the displacement vector is taken into account. Indeed, even if a pixel belonging to an object is correctly labelled, we incur a high loss if the object's center position estimation is completely wrong. This is taken into account with the second term.

The density estimation function $\pi(\mathcal{Z})$, which generates the posterior distributions stored in the tree leaves, is different depending if we are at an internal node or at a leaf of the tree. In both cases it provides distributions that factorize in two marginal distributions, for the class labels and the displacement vector, respectively. The marginal over the class labels is always a discrete distribution providing the probability of drawing a sample of a given class from the set $\mathcal{Z}$. The difference is with respect to the marginal over the displacement vector. We have a point-wise and uni-modal distribution at the internal nodes, while we keep track of multiple modes at the leaves.

Let $q \in \mathbb{P}(\{0, 1\})$ be the marginal distribution over the class labels defined as $q(c) = |\mathcal{Z}_c|/|\mathcal{Z}|$, where $\mathcal{Z}_0$ and $\mathcal{Z}_1$ are the sets of background and foreground samples in $\mathcal{Z}$, respectively. At the internal node level $\pi(\mathcal{Z})$ returns a probability distribution $Q_n \in \mathbb{P}(\mathcal{Y})$ defined as

$$Q_n\left(c, \mathbf{d}\right) = q(c)\delta(\mathbf{d} - \mathbf{d}^*)\,.$$

Here, $\mathbf{d}^*$ represents the single point-wise mode of the marginal distribution with respect to $\mathbf{d}$ (*i.e.* the second term), which is determined in a way to minimize the loss $L(\mathcal{Z}|Q_n)$ over the training samples. A local solution of the minimization problem can be found by iterating the following procedure[2]

$$\mathbf{d}^* \leftarrow \sum_{(x,(c,\mathbf{d})) \in \mathcal{Z}_1} \mathbf{d}\, K_\sigma(\mathbf{d} - \mathbf{d}^*) \Big/ \sum_{(x,(c,\mathbf{d})) \in \mathcal{Z}_1} K_\sigma(\mathbf{d} - \mathbf{d}^*)\,.$$

At the leaf level, instead, $\pi(\mathcal{Z})$ returns a probability distribution $Q_l \in \mathbb{P}(\mathcal{Y})$ defined as:

$$Q_l\left(c, \mathbf{d}\right) = q(c) \sum_{(x,(c',\mathbf{d}')) \in \mathcal{Z}_1} \delta(\mathbf{d} - \mathbf{d}')/|\mathcal{Z}_1|\,.$$

Here, the second term, *i.e.* the marginal over $\mathbf{d}$, is uniform over the set of displacement vectors belonging to foreground samples reaching the leaf.

We define finally a novel type of split function, which performs a test by exploiting the contextual information. This test is particularly interesting because it allows to check whether two pixels are expected to belong to the same object instance. The new split function $\varphi^{(cs)}(\mathbf{u}, I|t, \mathbf{h}_1, \mathbf{h}_2, \tau)$ takes as input a sample $(\mathbf{u}, I) \in \mathcal{X}$ and it is parametrized by a CS decision tree $t \in \mathcal{T}$ that is used for generating the contextual information, by two relative displacement vectors $\mathbf{h}_1, \mathbf{h}_2 \in \mathbb{R}^2$ that identify the position of two pixels relative to $\mathbf{u}$ and by a threshold $\tau$. The definition of our context-sensitive split functions $\varphi^{(cs)}$ is as follows:

$$\varphi^{(cs)}(\mathbf{u}, I|t, \mathbf{h}_1, \mathbf{h}_2, \tau) = \mathbb{1} \left[ \mathbb{E}_{(c,\mathbf{d},c',\mathbf{d}') \sim P_1 \cdot P_2} \left[ \mathbb{1} \left[ (c, c') = (1, 1) \right] K_\sigma(\mathbf{d} - \mathbf{d}') \right] < \tau \right] \qquad (8)$$

where $P_j = P(\cdot|(\mathbf{u} + \mathbf{h}_j, I), t)$, with $j = 1, 2$, are the posterior probabilities obtained from tree $t$ given samples at position $\mathbf{u}+\mathbf{h}_1$ and $\mathbf{u}+\mathbf{h}_2$ of image $I$, respectively. Please note that this test should not be confused with the regression split criterion in [9], which tries to partition the training set in a way to group examples with similar voting direction and length. Besides the novel context-sensitive split function we employ also standard split functions performing tests on $\mathcal{X}$ as defined in [24].

## 4    Experiments

To assess our proposed approach, we have conducted several experiments on the task of pedestrian detection. Detecting pedestrians is very challenging for Hough-voting based methods as they typically exhibit strong articulations of feet and arms, yielding to non-distinctive hypotheses in the Hough space. We evaluated our method on the TUD pedestrian data base [2] in two different ways: First, we show our detection results with training according to the standard protocol using 400 training images (where each image contains a single annotation of a pedestrian) and evaluation on the *Campus* and *Crossing* scenes, respectively (Section 4.1). With this experiment we show the improvement over state-of-the-art approaches when learning can be performed with simultaneous knowledge about context information. In a second variation (Section 4.2), we use the images of the Crossing scene (201 images) as a training set. Most images of this scene contain more than four persons with strong overlap and mutual occlusions. However, instead of using the original annotation which covers only pedestrians with at least 50% overlap (1008 bounding boxes), we use the more accurate, pixel-wise ground truth annotations of [23] for the entire scene that includes all persons and consists of 1215 bounding boxes. Please note that this annotation is even more detailed than the one presented in [4] with 1018 bounding boxes. The purpose of the second experiment is to show that our context-sensitive forest can exploit the availability of multiple training instances significantly better than state-of-the-art.

The most related work and therefore also the baseline in our experiments is the Hough Forest [9]. To guarantee a fair comparison, we use the same training parameters for [9] and our context sensitive forest: We trained 20 trees and the training data (including horizontally flipped images) was sampled homogeneously per category per image. The patch size was fixed to $30 \times 30$ and we performed 1600 node tests for finding the best split function parameters per node. The trees were stopped growing when $< 7$ samples were available. As image features, we used the the first 16 feature channels provided in the publicly available Hough Forest code of [9]. In order to obtain the object detection hypotheses from the Hough space, we use the same Non-maximum suppression (NMS) technique in all our experiments as suggested in [9]. To evaluate the obtained hypotheses, we use the standard PASAL-VOC criterion which requires the mutual overlap between ground truth and detected bounding boxes to be $\geq 50\%$. The additional parameter of (7) was fixed to $\sigma = 7$.

### 4.1    Evaluation using standard protocol training set

The standard training set contains 400 images where each image comes with a single pedestrian annotation. For our experiments, we rescaled the images by a factor of $0.5$ and doubled the training image set by including also the horizontally flipped images. We randomly chose 125 training samples per image for foreground and background, resulting in $2 \cdot 400 \cdot 2 \cdot 125 = 200k$ training samples per tree. For additional comparisons, we provide the results presented in the recent work on joint object detection and segmentation of [23], from which we also provide evaluation results of the Implicit Shape Model (ISM) [16]. However, please note that the results of [23] are based on a different baseline implementation. Moreover, we show the results of [4] when using the provided code and configuration files from the first authors homepage. Unfortunately, we could not reproduce the results of the original paper.

First, we discuss the results obtained on the Campus scene. This data set consists of 71 images showing walking pedestrians at severe scale differences and partial occlusions. The ground truth we use has been released with [4] and contains a total number of 314 pedestrians. Figure 3, first row, plot 1 shows the precision-recall curves when using 3 scales (factors 0.3, 0.4, 0.55) for our baseline [9] (blue), results from re-evaluating [4] (cyan, 5 scales), [23] (green) and our Context-Sensitive Forest without and with using the priority queue based tree construction (red/magenta). In case of not using the priority queue, we trained the trees according to a breadth-first way. We obtain a performance boost of $\approx 6\%$ in recall at a precision of $90\%$ when using both, context information and the priority based construction of our forest. The second plot in the first row of Figure 3 shows the results when the same forests are tested on the Crossing scene, using the more detailed ground

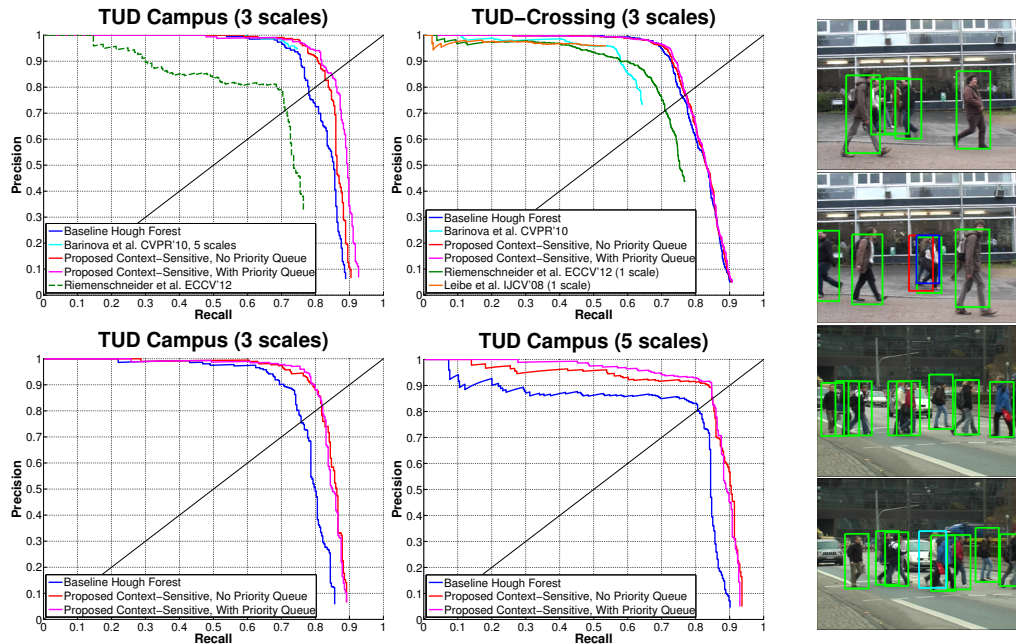

Figure 3: Precision-Recall Curves for detections, Top row: Standard training (400 images), evaluation on Campus and Crossing (3 scales). Bottom row: Training on Crossing annotations of [23], evaluation on Campus, 3 and 5 scales. Right images: Qualitative examples for Campus (top 2) and Crossing (bottom 2) scenes. (green) correctly found by our method (blue) ground truth (red) wrong association (cyan) missed detection.

truth annotations. The data set shows walking pedestrians (Figure 3, right side, last 2 images) with a smaller variation in scale compared to the Campus scene but with strong mutual occlusions and overlaps. The improvement with respect to the baseline is lower ($\approx 2\%$ gain at a precision of $90\%$) and we find similar developments of the curves. However, this comes somewhat expectedly as the training data does not properly reflect the occlusions we actually want to model.

### 4.2 Evaluation on Campus scene using Crossing scene as training set

In our next experiment we trained the forests (same parameters) on the novel annotations of [23] for the Crossing scene. Please note that this reduces the training set to only 201 images (we did not include the flipped images). Qualitative detection results are shown in Figure 3, right side, images 1 and 2. From the first precison-recall curve in the second row of Figure 3 we can see, that the margin between the baseline and our proposed method could be clearly improved (gain of $\approx 9\%$ recall at precision $90\%$) when evaluating on the same 3 scales. With evaluation on 5 scales (factors 0.34, 0.42, 0.51, 0.65, 0.76) we found a strong increase in the recall, however, at the cost of loosing $2-3\%$ of precision below a recall of $60\%$, as illustrated in the second plot of row 2 in Figure 3. While our method is able to maintain a precision above $90\%$ up to a recall of $\approx 83\%$, the baseline implementation drops already at a recall of $\approx 20\%$.

## 5 Conclusions

In this work we have presented Context-Sensitive Decision Forests with application to the object detection problem. Our new forest has the ability to access intermediate prediction (classification and regression) information about all samples of the training set and can therefore learn from contextual information throughout the growing process. This is in contrast to existing random forest methods used for object detection which typically treat training samples in an independent manner. Moreover, we have introduced a novel splitting criterion together with a mode isolation technique, which allows us to (a) perform a priority-driven way of tree growing and (b) install novel context-based test functions to check for mutual object centroid agreements. In our experimental results on pedestrian detection we demonstrated superior performance with respect to state-of-the-art methods and additionally found that our new algorithm can significantly better exploit training data containing multiple training objects.

**Acknowledgements.** Peter Kontschieder acknowledges financial support of the Austrian Science Fund (FWF) from project 'Fibermorph' with number P22261-N22.

## Footnotes

[1] we use the term predictor because we will jointly consider classification and regression.

[2]In the experiments conducted, we never exceeded 10 iterations for finding a mode.

# References

[1] Y. Amit and D. Geman. Shape quantization and recognition with randomized trees. *Neural Computation*, 1997.

[2] M. Andriluka, S. Roth, and B. Schiele. People-tracking-by-detection and people-detection-by-tracking. In *(CVPR)*, 2008.

[3] D. H. Ballard. Generalizing the hough transform to detect arbitrary shapes. *Pattern Recognition*, 13(2), 1981.

[4] O. Barinova, V. Lempitsky, and P. Kohli. On detection of multiple object instances using hough transforms. In *(CVPR)*, 2010.

[5] A. Bosch, A. Zisserman, and X. Muñoz. Image classification using random forests and ferns. In *(ICCV)*, 2007.

[6] L. Breiman. Random forests. In *Machine Learning*, 2001.

[7] A. Criminisi, J. Shotton, and E. Konukoglu. Decision forests: A unified framework for classification, regression, density estimation, manifold learning and semi-supervised learning. In *Foundations and Trends in Computer Graphics and Vision*, volume 7, pages 81–227, 2012.

[8] A. Criminisi, J. Shotton, D. Robertson, and E. Konukoglu. Regression forests for efficient anatomy detection and localization in CT scans. In *MICCAI-MCV Workshop*, 2010.

[9] J. Gall and V. Lempitsky. Class-specific hough forests for object detection. In *(CVPR)*, 2009.

[10] J. Gall, A. Yao, N. Razavi, L. Van Gool, and V. Lempitsky. Hough forests for object detection, tracking, and action recognition. *(PAMI)*, 2011.

[11] P. Geurts, D. Ernst, and L. Wehenkel. Extremely randomized trees. *Machine Learning*, 2006.

[12] R. Girshick, J. Shotton, P. Kohli, A. Criminisi, and A. Fitzgibbon. Efficient regression of general-activity human poses from depth images. In *(ICCV)*, 2011.

[13] T. Hastie, R. Tibshirani, and J. H. Friedman. *The Elements of Statistical Learning*. Springer, 2009.

[14] R. A. Hummel and S. W. Zucker. On the foundations of relaxation labeling. *(PAMI)*, 5(3):267–287, 1983.

[15] P. Kontschieder, S. Rota Bulò, H. Bischof, and M. Pelillo. Structured class-labels in random forests for semantic image labelling. In *(ICCV)*, 2011.

[16] B. Leibe, A. Leonardis, and B. Schiele. Robust object detection with interleaved categorization and segmentation. *(IJCV)*, 2008.

[17] R. Marée, P. Geurts, J. Piater, and L. Wehenkel. Random subwindows for robust image classification. In *(CVPR)*, 2005.

[18] A. Montillo, J. Shotton, J. Winn, J. E. Iglesias, D. Metaxas, and A. Criminisi. Entangled decision forests and their application for semantic segmentation of CT images. In *(IPMI)*, 2011.

[19] F. Moosmann, B. Triggs, and F. Jurie. Fast discriminative visual codebooks using randomized clustering forests. In *(NIPS)*, 2006.

[20] M. Pelillo and M. Refice. Learning compatibility coefficients for relaxation labeling processes. *(PAMI)*, 16(9):933–945, 1994.

[21] A. Rabinovich, A. Vedaldi, C. Galleguillos, E. Wiewiora, and S. Belongie. Objects in context. In *(ICCV)*, 2007.

[22] N. Razavi, J. Gall, and L. Van Gool. Scalable multi-class object detection. In *(CVPR)*, 2011.

[23] H. Riemenschneider, S. Sternig, M. Donoser, P. M. Roth, and H. Bischof. Hough regions for joining instance localization and segmentation. In *(ECCV)*, 2012.

[24] J. Shotton, M. Johnson, and R. Cipolla. Semantic texton forests for image categorization and segmentation. In *(CVPR)*, 2008.

[25] Z. Tu. Auto-context and its application to high-level vision tasks. In *(CVPR)*, 2008.

[26] O. Woodford, M. Pham, A. Maki, F. Perbet, and B. Stenger. Demisting the hough transform for 3d shape recognition and registration. In *(BMVC)*, 2011.

